# A Boundary Hunting Radial Basis Function Classifier Which Allocates Centers Constructively

**Eric I. Chang and Richard P. Lippmann**
MIT Lincoln Laboratory
Lexington, MA 02173-0073, USA

## Abstract

A new boundary hunting radial basis function (BH-RBF) classifier which allocates RBF centers constructively near class boundaries is described. This classifier creates complex decision boundaries only in regions where confusions occur and corresponding RBF outputs are similar. A predicted square error measure is used to determine how many centers to add and to determine when to stop adding centers. Two experiments are presented which demonstrate the advantages of the BH-RBF classifier. One uses artificial data with two classes and two input features where each class contains four clusters but only one cluster is near a decision region boundary. The other uses a large seismic database with seven classes and 14 input features. In both experiments the BH-RBF classifier provides a lower error rate with fewer centers than are required by more conventional RBF, Gaussian mixture, or MLP classifiers.

## 1   INTRODUCTION

Radial basis function (RBF) classifiers have been successfully applied to many pattern classification problems (Broomhead, 1988, Ng, 1991). These classifiers have the advantages of short training times and high classification accuracy. In addition, RBF outputs estimate minimum-error Bayesian *a posteriori* probabilities (Richard, 1991). Performing classification with RBF outputs requires selecting the output which is highest for each input. In regions where one class dominates, the Bayesian *a posteriori* probability for that class will be uniformly "high" and near 1.0. Detailed modeling of the variation of the Bayesian *a posteriori* probability in these regions is not necessary for classification. Only

at the boundary between different classes is accurate estimation of the Bayesian *a posteriori* probability necessary for high classification accuracy. If the boundary between different classes can be located in the input space, RBF centers can be judiciously allocated in those regions without wasting RBF centers in regions where accurate estimation of the Bayesian *a posteriori* probability does not improve classification performance.

In general, having more RBF centers allows better approximation of the desired output. While training a RBF classifier, the number of RBF centers must be selected. The traditional approach has been to randomly choose patterns from the training set as centers, or to perform $K$-means clustering on the data and then to use these centers as the RBF centers. Frequently the correct number of centers to use is not known *a priori* and the number of centers has to be tuned. Also, with $K$-means clustering, the centers are distributed without considering their usefulness in classification. In contrast, a constructive approach to adding RBF centers based on modeling Bayesian *a posteriori* probabilities accurately only near class boundaries provides good performance with fewer centers than are required to separately model class PDF's.

Many algorithms have been proposed for constructively building up the structure of a RBF network (Mel, 1991). However, the algorithms proposed have all been designed for training a RBF network to perform function mapping. For mapping tasks, accuracy is important throughout the input region and the mean squared error is the criterion that is minimized. In classification tasks, only boundaries between different classes are important and the overall mean squared error is not as important as the error in class boundaries.

## 2   ALGORITHM DESCRIPTION

A block diagram of a new boundary hunting RBF (BH-RBF) classifier that adds centers constructively near class boundaries is presented in Figure 1. A simple unimodal Gaussian classifier is first formed by clustering the training patterns from a randomly selected class and assigning a center to that class. The confusion matrix generated by using this simple classifier is then examined to determine the pair of classes $A$ and $B$, which have the most mutual confusion. Training patterns that are close to the boundary between these two classes are determined by looking at the outputs of the RBF classifier. Boundary patterns

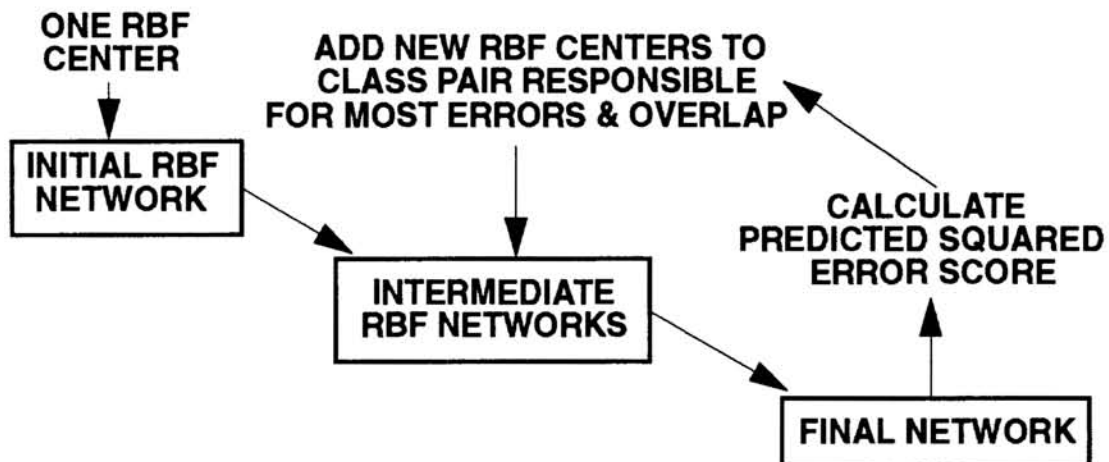

Figure 1: Block Diagram of Training of BH-RBF Network

which produce similar "high" outputs for both classes that are different by less than a "closecall" threshold are used to produce new cluster centers.

Figure 2 shows RBF outputs corresponding to class $A$ and $B$ as the input varies over a small range. This figure illustrates how network outputs are used to determine the "close-call" region between classes. Network outputs are high in regions dominated by a particular class and therefore these regions are outside the boundary between different classes. Network outputs are close in the region where the absolute difference of the two highest network outputs is less than the closecall threshold. Training patterns which fall into this closecall region plus all the points that are misclassified as the other class in the class pair are considered to be points in the boundary. For example, a pattern in class $A$ which is misclassified as class $B$ would be considered to be in the boundary between class $A$ and $B$. On the other hand, a pattern in class $A$ which is misclassified as class $C$ would not be placed in the boundary between class $A$ and $B$.

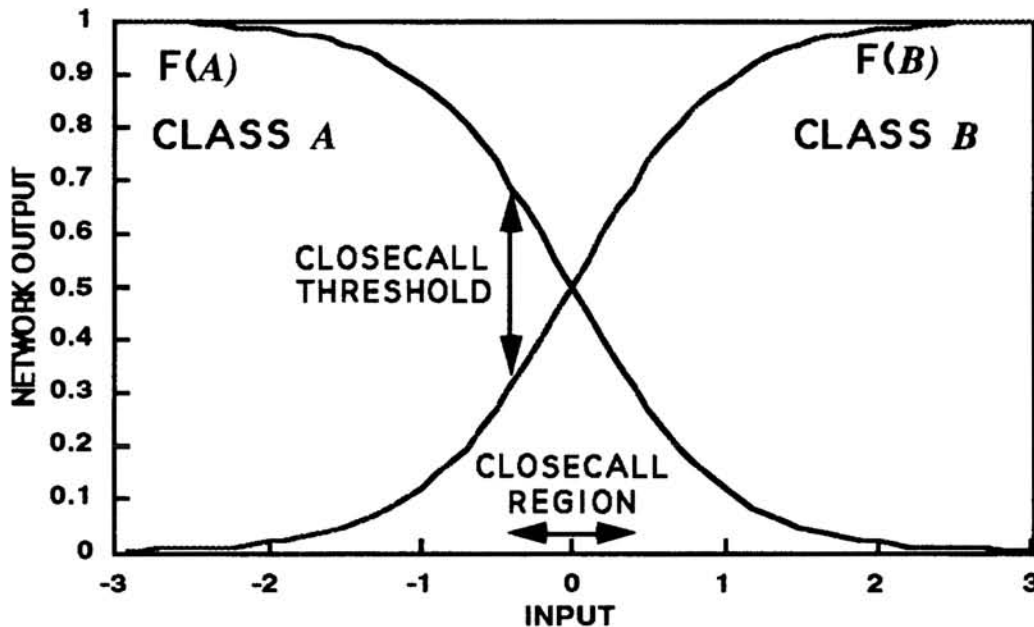

Figure 2: Using the Network Output to Determine Closecall Regions

After the patterns which belong in the boundary are determined, clustering is performed separately on boundary patterns from different classes using $K$-means clustering and a number of centers ranging from zero to a preset maximum number of centers. After the centers are found, new RBF classifiers are trained using the new sets of centers plus the original set of centers. The combined set of centers that provides the best performance is saved and the cycle repeats again by finding the next class pair which accounts for the most remaining confusions. Overfitting by adding too many centers at a time is avoided by using the predicted squared error (PSE) as the criterion for choosing new centers (Barron, 1984):

$$PSE = RMS + \frac{C \times \sigma^2}{N} .$$

In this equation, *RMS* is the root mean squared error on the training set, $\sigma^2$ estimates the variance of the error, $C$ is the total number of centers in the RBF classifier, and $N$ is the total number of patterns in the training set. The error variance $\sigma^2$ is selected empirically using left-out evaluation data. Different values of $\sigma^2$ are tried and the value which provides the best performance on the evaluation data is chosen. On each cycle, different number of centers are tried for each class of the selected class pair and the PSE is used to select the best subset of centers. The best PSE on each cycle is used to determine when training should be stopped to prevent overfitting. Training stops after the PSE has not decreased for five consecutive cycles.

## 3   EXPERIMENTAL RESULTS

Two experiments were performed using the new BH-RBF classifier, a more conventional RBF classifier, a Gaussian mixture classifier (Ng, 1991), and a MLP classifier. Five regular RBF classifiers (RBF) were trained by assigning 1, 2, 3, 4, or 5 centers to each class. Similarly, five Gaussian mixture classifiers (GMIX) were trained with 1, 2, 3, 4, or 5 centers in each class. The means of each center were trained individually using *K*-means clustering to find the centers for patterns from each class. The diagonal covariance of each center was set using all the patterns that were assigned to a cluster during the last pass of *K*-means clustering. The structure of the regular RBF classifier and the Gaussian mixture classifier are identical when the number of centers are the same. The only difference between the classifiers is the method used to train parameters.

MLP classifiers were trained for 10 independent trials for each data set. The number of hidden nodes was varied from 2 to 30 in increments of 2. The goal of the experiment was to explore the relationship between the complexity of the classifier and the classification accuracy of the classifier. Training was stopped using cross validation to avoid overfitting.

### 3.1 FOUR-CLUSTER DATABASE

The first problem is an artificial data set designed to illustrate the difference between BH-RBF and other classifiers. There are two classes, each class consist of one large Gaussian cluster with 700 random points and three smaller clusters with 100 points each. Figure 3 shows the distribution of the data and the ideal decision boundary if the actual centers and variances are used to train a Bayesian minimum error classifier. There were 2000 training patterns, 2000 evaluation patterns, and 2000 test patterns. The BH-RBF classifier was trained with the closecall threshold set to 0.75, $\sigma^2$ set to 0.5, and a maximum of two extra centers per class at between each pair of classes. The theoretically optimal Bayesian classifier for this database provides the error rate of 1.95% on the test set. This optimal Bayesian classifier is obtained using the actual centers, variances, and *a priori* probability used to generate the data in a Gaussian mixture classifier. In a real classification task, these center parameters are not known and have to be estimated from training data.

Figure 4 shows the testing error rate of the three different classifiers. The BH-RBF classifier was able to achieve 2.35% error rate with only 5 centers and the error rate gradually decreased to 2.15% with 15 centers. The BH-RBF classifier performed well with few centers because it allocated these centers near the boundary between the two classes. On the other hand, the performance of the RBF classifier and the Gaussian mixture classifier was worse with few centers. These classifiers performed worse because they allocated centers

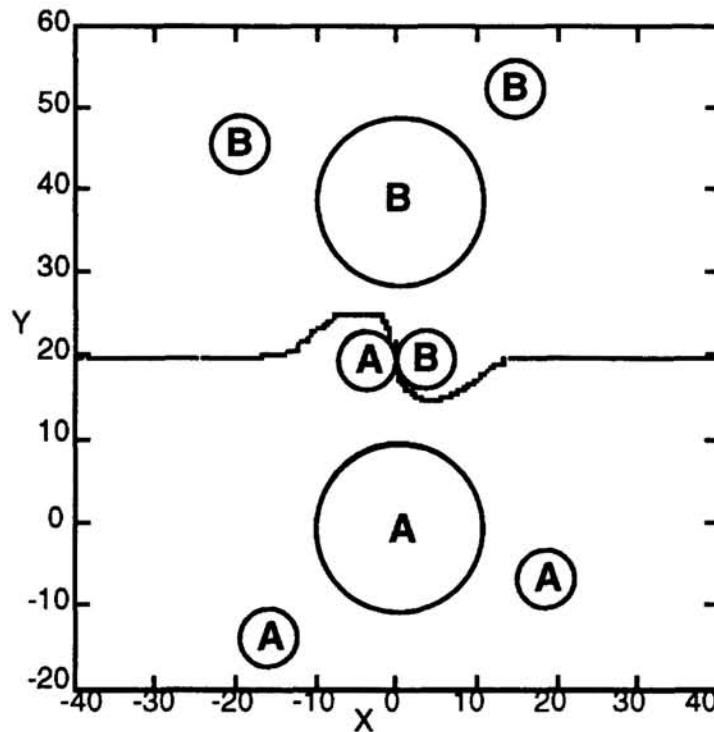

Figure 3: The Artificially Generated Four-Cluster Problem

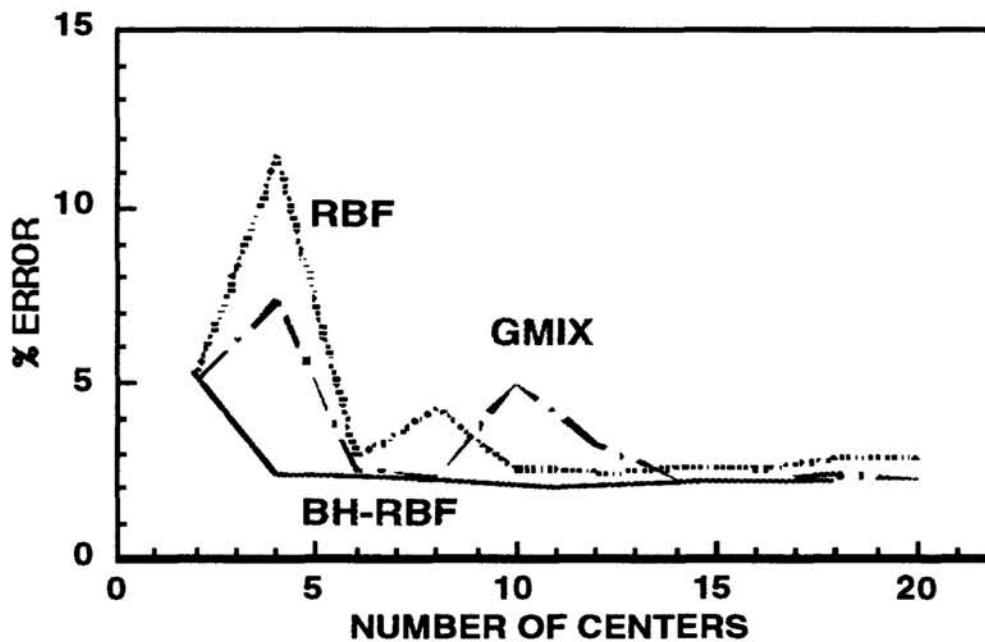

Figure 4: Testing Error Rate Of The BH-RBF Classifier, The Gaussian Mixture Classifier, And The Regular RBF Classifier On The Four-Cluster Problem.

in regions that had many patterns. The training algorithm did not distinguish between patterns that are easily confusable between classes (i.e. near the class boundary) and patterns that clearly belong in a given class. Furthermore, adding more centers did not monotoni-

cally decrease the error rate. For example, the RBF classifier had 5% error using two centers, but when the number of centers was increased to four, the error rate jumped to 11%. Only until the number of centers increased above 14 did the RBF classifier and the Gaussian mixture classifier's error rates converge. The RBF and the Gaussian mixture classifiers performed poorly with few centers because the centers were concentrated away from the decision boundary due to the high concentration of data far away from the boundary. Thus, there weren't enough centers to model the decision boundary accurately. The BH-RBF classifier added centers near the boundary and thus was able to define an accurate boundary with fewer centers.

Figure 5 presents the results from training MLP classifiers on the same data set using different numbers of hidden nodes. The learning rate was set to 0.001, the momentum term was set to 0.6, and each classifier was trained for 100 epochs. The error rate on a left out evaluation set was checked to assure that the net had not overfitted the training data. As the number of hidden nodes increased, the MLP classifier generally performed better. However, the testing error rate did not decrease monotonically as the number of hidden nodes increased. Furthermore, the random initial condition set by the different random seeds affected the classification error rate of each classifier. In comparison, the training algorithms used for BH-RBF, RBF, and GMIX classifiers do not exhibit such sensitivity to initial conditions.

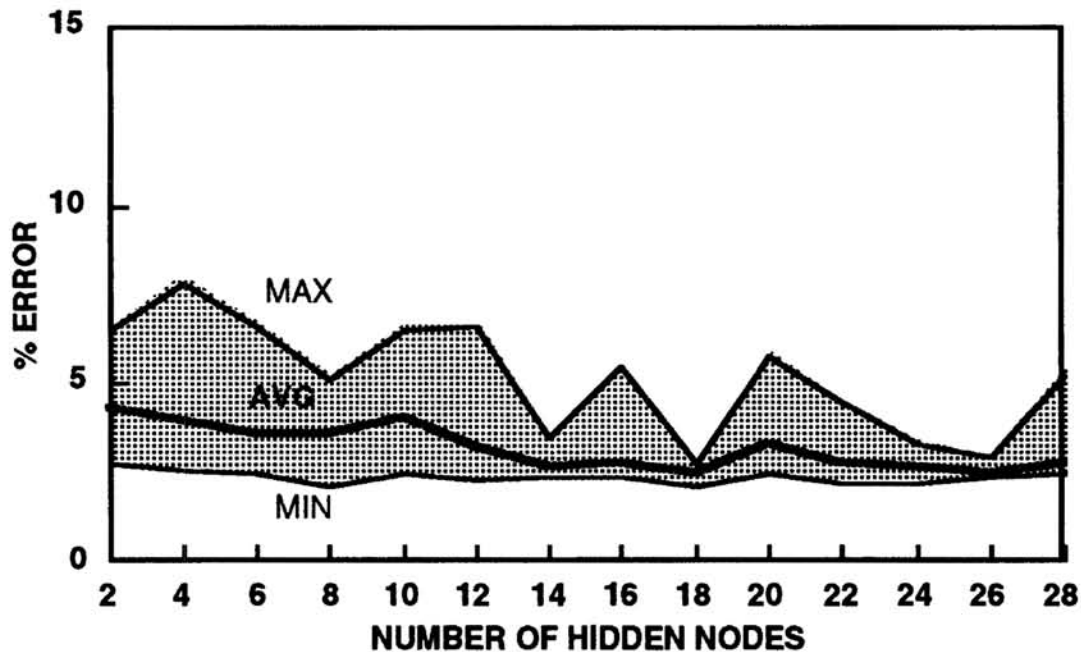

Figure 5: Testing Error Rate Of The MLP Classifiers On The Four-Cluster Problem

## 3.2  SEISMIC DATABASE

The second problem consists of data for classification of seismic events. The input consist of 14 continuous and binary measurements derived from seismic waveform signals. These features are used to classify a waveform as belonging to one of 7 classes which represent different seismic phases. There were 3038 training, 3033 evaluation, and 3034 testing pat-

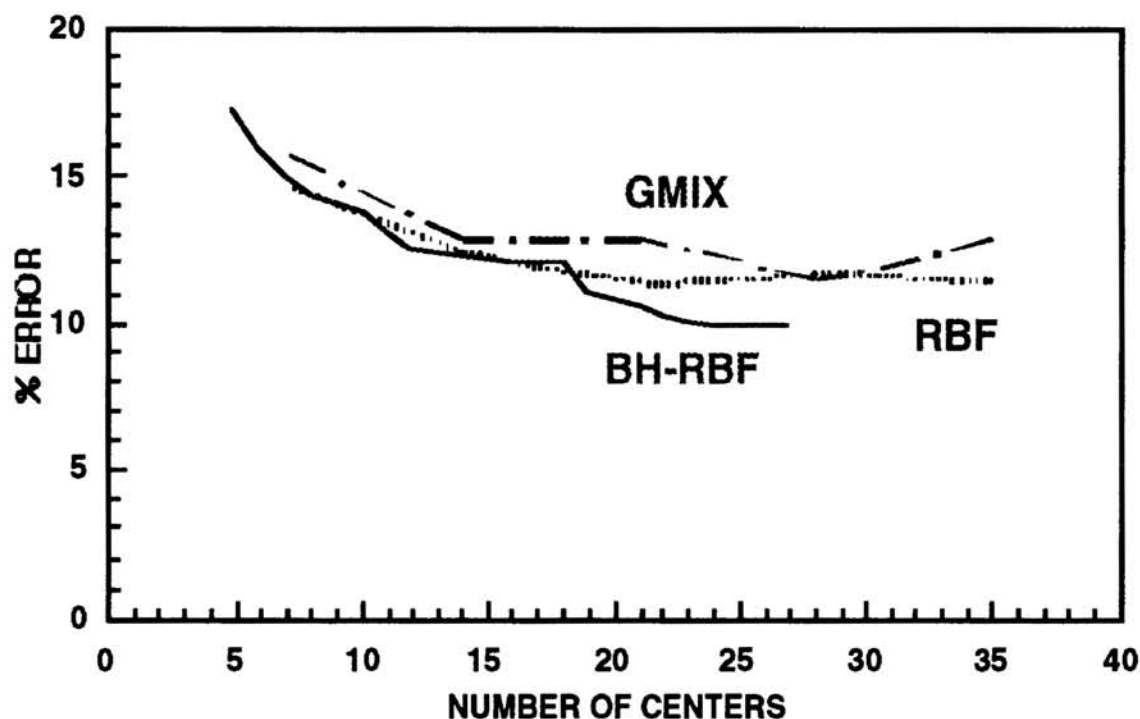

Figure 6: Error Rate Comparison Between The BH-RBF Classifier, The Regular
RBF Classifier, And The Gaussian Mixture Classifier On The Seismic Problem

terns. Once again, the number of centers per class was varied from 1 to 5 for the regular
RBF classifier and the Gaussian mixture classifier, while the BH-RBF classifier was
started with 1 center in the first class and then more centers were automatically assigned.
The BH-RBF classifier was trained with the closecall threshold set to 0.75, $\sigma^2$ set to 0.5,
and a maximum of one extra center per class at each boundary. The parameters were cho-
sen according to the performance of the classifier on the left-out evaluation data. For this
problem, the closecall threshold and $\sigma^2$ turned out to be the same as the ones used in the
four-cluster problem.

Figure 6 shows the error rate on the testing patterns for all three classifiers. The BH-RBF
classifier clearly performed better than the regular RBF classifier and the Gaussian mix-
ture classifier. The BH-RBF classifier added centers only at the boundary region where
they improved discrimination. Also, the diagonal covariance of the added centers are more
local in their influence and can improve discrimination of a particular boundary without
affecting other decision region boundaries.

MLP classifiers were also trained on this data set with the number of hidden nodes varying
from 2 to 32 in increments of 2. The learning rate was set to 0.001, the momentum term
was set to 0.6, and each classifier was trained for 100 epochs. The classification error rate
on the left-out evaluation set showed that the network had not overfitted on the training
data. Once more, the MLP classifiers exhibited great sensitivity to initial conditions, espe-
cially when the number of hidden nodes were small. Also, for this high dimensionality
classification task, even the best performance of the MLP classifier (15.5%) did not match
the best performance of the BH-RBF classifier. This result suggests that for this high

dimensionality data, the radially symmetric boundaries formed with local basis functions such as the RBF classifier are more appropriate than the ridge-like boundaries formed with the MLP classifier.

# 4   CONCLUSION

A new boundary-hunting RBF classifier was developed which adds RBF centers constructively near boundaries of classes which produce classification confusions. Experimental results from two problems differing in input dimension, number of classes, and difficulty show that the BH-RBF classifier performed better than traditional training algorithms used for RBF, Gaussian mixture, and MLP classifiers. Experiments have also been conducted on other problems such as Peterson and Barney's vowel database and the disjoint database used by Ng (Peterson, 1952, Ng, 1990). In all experiments, the BH-RBF constructive algorithm performed at least as well as the traditional RBF training algorithm. These results, and the experiments described above, confirm the hypothesis that better discrimination performance can be achieved by training a classifier to perform discrimination instead of probability density function estimation.

**Acknowledgments**

This work was supported by DARPA. The views expressed are those of the authors and do not reflect the official policy or position of the U.S. Government. Experiments were conducted using *LNKnet*, a general purpose classifier program developed at Lincoln Laboratory by Richard Lippmann, Dave Nation, and Linda Kukolich.

**References**

G. E. Peterson and H. L. Barney. (1952) Control Methods Used in a Study of Vowels. *The Journal of the Acoustical Society of America* **24:2**, 175-84.

A. Barron. (1984) Predicted squared error: a criterion for automatic model selection.   In S. Farlow, Editor. *Self-Organizing Methods in Modeling*. New York, Marcel Dekker.

D. S. Broomhead and D. Lowe. (1988) *Radial Basis Functions, multi-variable functional interpolation and adaptive networks*. Technical Report RSRE Memorandum No. 4148, Royal Speech and Radar Establishment, Malvern, Worcester, Great Britain.

B. W. Mel and S. M. Omohundro. (1991) How Receptive Field Parameters Affect Neural Learning. In R. Lippmann, J. Moody and D. Touretzky (Eds.), *Advances in Neural Information Processing Systems 3*, 1991. San Mateo, CA: Morgan Kaufman.

K. Ng and R. Lippmann. (1991) A Comparative Study of the Practical Characteristics of Neural Networks and Conventional Pattern Classifiers. In R. Lippmann, J. Moody and D. Touretzky (Eds.), *Advances in Neural Information Processing Systems 3*, 1991. San Mateo, CA: Morgan Kaufman.

M.D. Richard and R. P. Lippmann. (1991) Neural Network Classifier Estimates Bayesian *a posteriori* Probabilities. *Neural Computation*, Volume 3, Number 4.